# Multiple Cause Vector Quantization

**David A. Ross and Richard S. Zemel**
Department of Computer Science
University of Toronto
{dross,zemel}@cs.toronto.edu

## Abstract

We propose a model that can learn parts-based representations of high-dimensional data. Our key assumption is that the dimensions of the data can be separated into several disjoint subsets, or factors, which take on values independently of each other. We assume each factor has a small number of discrete states, and model it using a *vector quantizer*. The selected states of each factor represent the *multiple causes* of the input. Given a set of training examples, our model learns the association of data dimensions with factors, as well as the states of each VQ. Inference and learning are carried out efficiently via variational algorithms. We present applications of this model to problems in image decomposition, collaborative filtering, and text classification.

## 1   Introduction

Many collections of data exhibit a common underlying structure: they consist of a number of parts or factors, each of which has a small number of discrete *states*. For example, in a collection of facial images, every image contains eyes, a nose, and a mouth (except under occlusion), each of which has a range of different appearances. A specific image can be described as a *composite sketch*: a selection of the appearance of each part, depending on the individual depicted.

In this paper, we describe a stochastic generative model for data of this type. This model is well-suited to decomposing images into parts (it can be thought of as a Mr. Potato Head model), but also applies to domains such as text and collaborative filtering in which the parts correspond to latent features, each having several alternative instantiations. This representational scheme is powerful due to its combinatorial nature: while a standard clustering/VQ method containing $N$ states can represent at most $N$ items, if we divide the $N$ into $j$-state VQs, we can represent $j^{N/j}$ items. MCVQ is also especially appropriate for high-dimensional data in which many values may be unspecified for a given input case.

## 2   Generative Model

In MCVQ we assume there are $K$ factors, each of which is modeled by a vector quantizer with $J$ states. To generate an observed data example of $D$ dimensions, $\mathbf{x} \in \Re^D$, we stochastically select one state for each VQ, and one VQ for each dimension. Given these selections, a single state from a single VQ determines the value of each data dimension $x_d$.

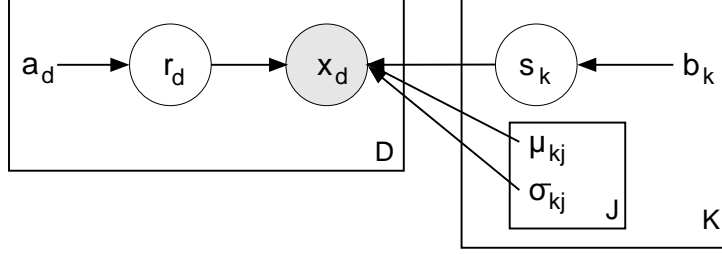

Figure 1: Graphical model representation of MCVQ. We let $\mathbf{r}_{d=1}$ represent all the variables $r_{d=1,k}$, which together select a VQ for $x_1$. Similarly, $\mathbf{s}_{k=1}$ represents all $s_{k=1,j}$, which together select a state of VQ 1. The plates depict repetitions across the appropriate dimensions for each of the three variables: the $K$ VQs, the $J$ states (codebook vectors) per VQ, and the $D$ input dimensions.

The selections are represented as binary latent variables, $S = \{s_{kj}\}, R = \{r_{dk}\}$, for $d = 1...D, k = 1...K$, and $j = 1...J$. The variable $s_{kj} = 1$ if and only if state $j$ has been selected from VQ $k$. Similarly $r_{dk} = 1$ when VQ $k$ has been selected for data dimension $d$. These variables can be described equivalently as multinomials, $\mathbf{s}_k \in 1...J, \mathbf{r}_d \in 1...K$; their values are drawn according to their respective priors, $\mathbf{a}_k$ and $\mathbf{b}_d$. The graphical model representation of MCVQ is given in Fig. 1.

Assuming each VQ state specifies the mean as well as the standard deviation of a Gaussian distribution, and the noise in the data dimensions is conditionally independent, we have (where $\theta = \{\mu_{dkj}, \sigma_{dkj}\}$):

$$P(\mathbf{x}|R, S, \theta) = \prod_d \prod_{k,j} \mathcal{N}(x_d \, ; \, \mu_{dkj}, \, \sigma_{dkj})^{r_{dk} \, s_{kj}}$$

The resulting model can be thought of as a two-dimensional mixture model, in which $J * K$ possible states exist for each data dimension ($x_d$). The selections of states for the different data dimensions are joined along the $J$ dimension and occur independently along the $K$ dimension.

## 3   Learning and Inference

The joint distribution over the observed vector $\mathbf{x}$ and the latent variables is

$$P(\mathbf{x}, R, S|\theta) = P(R|\theta)P(S|\theta)P(\mathbf{x}|R, S, \theta) = \prod_{d,k} a_{dk}^{r_{dk}} \prod_{k,j} b_{kj}^{s_{kj}} \prod_{d,k,j} \mathcal{N}(x_d \, ; \, \theta)^{r_{dk}s_{kj}}$$

Given an input $\mathbf{x}$, the posterior distribution over the latent variables, $P(R, S|\mathbf{x}, \theta)$, cannot tractably be computed, since all the latent variables become dependent.

We apply a variational EM algorithm to learn the parameters $\theta$, and infer hidden variables given observations. We approximate the posterior distribution using a factored distribution, where $g$ and $m$ are variational parameters related to $r$ and $s$ respectively:

$$Q(R, S|\mathbf{x}, \theta) = \left( \prod_{d,k} g_{dk}^{r_{dk}} \right) \left( \prod_{k,j} m_{kj}^{s_{kj}} \right)$$

The variational free energy, $\mathcal{F}(Q, \theta) = E_Q \big[ - \log P(\mathbf{x}, R, S|\theta) + \log Q(R, S|\mathbf{x}, \theta) \big]$ is:

$$\mathcal{F} = E_Q \Big[ \sum_{d,k} r_{dk} \log(g_{dk}/a_{kj}) + \sum_{k,j} s_{kj} \log(m_{kj}/b_{kj}) + \sum_{d,k,j} r_{dk}s_{kj} \log \mathcal{N}(x_d \, ; \, \theta) \Big]$$

$$= \sum_{k,j} m_{kj} \log m_{kj} + \sum_{d,k} g_{dk} \log g_{dk} + \sum_{d,k,j} g_{dk} \, m_{kj} \, \epsilon_{dkj}$$

where $\epsilon_{dkj} = \log \sigma_{dkj} + \frac{(x_d - \mu_{dkj})^2}{2\sigma_{dkj}^2}$, and we have assumed uniform priors for the selection variables. The negative of the free energy $-\mathcal{F}$ is a lower bound on the log likelihood of generating the observations. The variational EM algorithm improves this bound by iteratively improving $-\mathcal{F}$ with respect to $Q$ (E-step) and to $\theta$ (M-step).

Let $C$ be the set of training cases, and $Q^c$ be the approximation to the posterior distribution over latent variables given the training case (observation) $c \in C$. We further constrain this variational approach, forcing the $\{g_{dk}^c\}$ to be consistent across all observations $\mathbf{x}^c$. Hence these parameters relating to the gating variables that govern the selection of a factor for a given observation dimension, are not dependent on the observation. This approach encourages the model to learn representations that conform to this constraint. That is, if there are several posterior distributions consistent with an observed data vector, it favours distributions over $\{\mathbf{r}_d\}$ that are consistent with those of other observed data vectors. Under this formulation, only the $\{m_{kj}^c\}$ parameters are updated during the E step for each observation $c$:

$$m_{kj}^c = \exp\left(-\sum_d g_{dk} \, \epsilon_{dkj}^c\right) \Big/ \sum_{\alpha=1}^{J} \exp\left(-\sum_d g_{dk} \, \epsilon_{d\alpha k}^c\right)$$

The M step updates the parameters, $\mu$ and $\sigma$, from each hidden state $kj$ to each input dimension $d$, and the gating variables $\{g_{dk}\}$:

$$g_{dk} = \exp\left(-\frac{1}{C}\sum_{c,j} m_{kj}^c \, \epsilon_{dkj}^c\right) \Big/ \sum_{\beta=1}^{K} \exp\left(-\frac{1}{C}\sum_{c,j} m_{j\beta}^c \, \epsilon_{dj\beta}^c\right)$$

$$\mu_{dkj} = \sum_c m_{kj}^c x_d^c \Big/ \sum_c m_{kj}^c \qquad \sigma_{dkj}^2 = \sum_c m_{kj}^c (x_d^c - \mu_{dkj})^2 \Big/ \sum_c m_{kj}^c$$

A slightly different model formulation restricts the selections of VQs, $\{r_{dk}\}$, to be the same for each training case. Variational EM updates for this model are identical to those above, except that the $\frac{1}{C}$ terms in the updates for $g_{dk}$ disappear. In practice, we obtain good results by replacing this $\frac{1}{C}$ term with an inverse temperature parameter, that is annealed during learning. This can be thought of as gradually moving from a generative model in which the $r_{dk}$'s can vary across examples, to one in which they are the same for each example.

The inferred values of the variational parameters specify a posterior distribution over the VQ states, which in turn implies a mixture of Gaussians for each input dimension. Below we use the mean of this mixture, $\hat{x_d^c} = \sum_{k,j} m_{kj}^c \, g_{dk} \, \mu_{dkj}$, to measure the model's reconstruction error on case $c$.

## 4 Related models

MCVQ falls into the expanding class of unsupervised algorithms known as *factorial methods*, in which the aim of the learning algorithm is to discover multiple independent causes, or factors, that can well characterize the observed data. Its direct ancestor is Cooperative Vector Quantization [1, 2, 3], which models each data vector as a linear combination of VQ selections. Another part-seeking algorithm, non-negative matrix factorization (NMF) [4], utilizes a non-negative linear combination of non-negative basis functions. MCVQ entails another round of competition, from amongst the VQ selections rather than the linear combination of CVQ and NMF, which leads to a division of input dimensions into separate causes. The contrast between these approaches mirrors the development of the competitive mixture-of-experts algorithm which grew out of the inability of a cooperative, linear combination of experts to decompose inputs into separable experts.

MCVQ also resembles a wide range of generative models developed to address image segmentation [5, 6, 7]. These are generally complex, hierarchical models designed to focus on a different aspect of this problem than that of MCVQ: to dynamically decide which pixels belong to which objects. The chief obstacle faced by these models is the unknown pose (primarily limited to position) of an object in an image, and they employ learned object models to find the single object that best explains each pixel. MCVQ adopts a more constrained solution w.r.t. part locations, assuming that these are consistent across images, and instead focuses on the assembling of input dimensions into parts, and the variety of instantiations of each part. The constraints built into MCVQ limit its generality, but also lead to rapid learning and inference, and enable it to scale up to high-dimensional data.

Finally, MCVQ also closely relates to sparse matrix decomposition techniques, such as the *aspect model* [8], a latent variable model which associates an unobserved class variable, the aspect $z$, with each observation. Observations consist of co-occurrence statistics, such as counts of how often a specific word occurs in a document. The latent Dirichlet allocation model [9] can be seen as a proper generative version of the aspect model: each document/input vector is not represented as a set of labels for a particular vector in the training set, and there is a natural way to examine the probability of some unseen vector. MCVQ shares the ability of these models to associate multiple aspects with a given document, yet it achieves this by sampling from multiple aspects in parallel, rather than repeated sampling of an aspect within a document. It also imposes the additional selection of an aspect for each input dimension, which leads to a soft decomposition of these dimensions based on their choice of aspect. Below we present some initial experiments examining whether MCVQ can match the successful application of the aspect model to information retrieval and collaborative filtering problems, after evaluating it on image data.

## 5 Experimental Results

### 5.1 Parts-based Image Decomposition: Shapes and Faces

The first dataset used to test our model consisted of $11 \times 11$ gray-scale images, as pictured in Fig. 2a. Each image in the set contains three shapes: a box, a triangle, and a cross. The horizontal position of each shape is fixed, but the vertical position is allowed to vary, uniformly and independently of the positions of the other shapes. A model containing 3 VQs, 5 states each, was trained on a set of 100 shape images. In this experiment, and all experiments reported herein, annealing proceeded linearly from an integer less than $C$ to 1. The learned representation, pictured in Fig. 2b, clearly shows the specialization of each VQ to one of the shapes.

The training set was selected so that none of the examples depict cases in which all three shapes are located near the top of the image. Despite this handicap, MCVQ is able to learn the full range of shape positions, and can accurately reconstruct such an image (Fig. 2c). In contrast, standard unsupervised methods such as Vector Quantization (Fig. 3a) and Principal Component Analysis (Fig. 3b) produce holistic representations of the data, in which each basis vector tries to account for variation observed across the entire image. Non-negative matrix factorization does produce a parts-based representation (Fig. 3c), but captures less of the data's structure. Unlike MCVQ, NMF does not group related parts, and its generative model does not limit the combination of parts to only produce valid images.

As an empirical comparison, we tested the reconstruction error of each of the aforementioned methods on an independent test set of 629 images. Since each method has one or more free parameters (e.g. the # of principal components) we chose to relate models with similar description lengths[1]. Using a description length of about $5.9 \times 10^5$ bits, and pixel

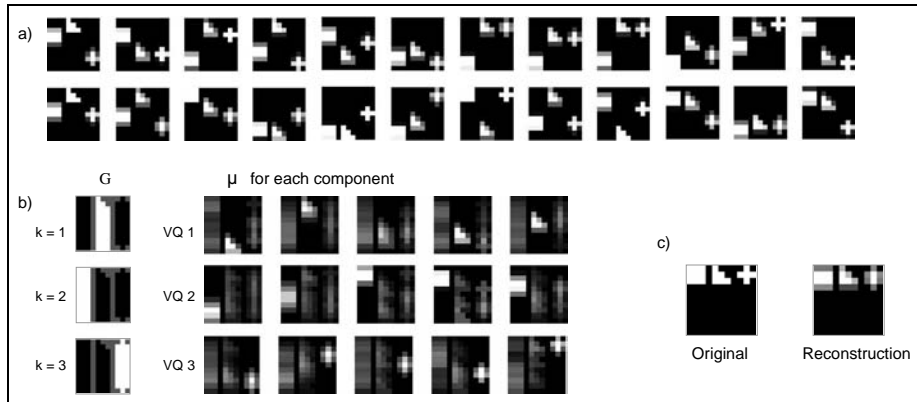

Figure 2: a) A sample of 24 training images from the Shapes dataset. b) A typical representation learned by MCVQ with 3 VQs and 5 states per VQ. c) Reconstruction of a test image: original (left) and reconstruction (right).

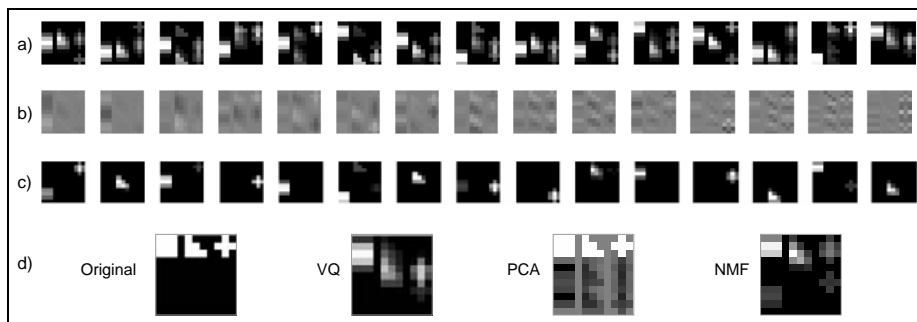

Figure 3: Other methods trained on shape images: a) VQ, b) PCA, and c) NMF. d) Reconstruction of a test image by the three methods (cf. Fig. 2c).

values ranging from -1 to 1, the average r.m.s. reconstruction error was 0.21 for MCVQ (3 VQs), 0.22 for PCA, 0.35 for NMF, and 0.49 for VQ. Note that this metric may be useful in determining the number of VQs, e.g., MCVQ with 6 VQs had an eror of 0.6.

As a more interesting visual application, we trained our model on a database of face images (www.ai.mit.edu/cbcl/projects).The dataset consists of $19 \times 19$ gray-scale images, each containing a single frontal or near-frontal face. A model of 6 VQs with 12 states each was trained on 2000 images, requiring 15 iterations of EM to converge. As with shape images, the model learned a parts-based representation of the faces.

The reconstruction of two test images, along with the specific parts used to generate each, is illustrated in Fig. 4. It is interesting to note that the pixels comprising a single part need not be physically adjacent (e.g. the eyes) as long as their appearances are correlated. We again compared the reconstruction error of MCVQ with VQ, PCA, and NMF. The training and testing sets contained 1800 and 629 images respectively. Using a description length of $1.5 \times 10^6$ bits, and pixel values ranging from -1 to 1, the average r.m.s. reconstruction error

---

number of bits to encode all the test examples using the model. This metric balances the large model cost and small encoding cost of VQ/MCVQ with the small model cost and large encoding cost of PCA/NMF.

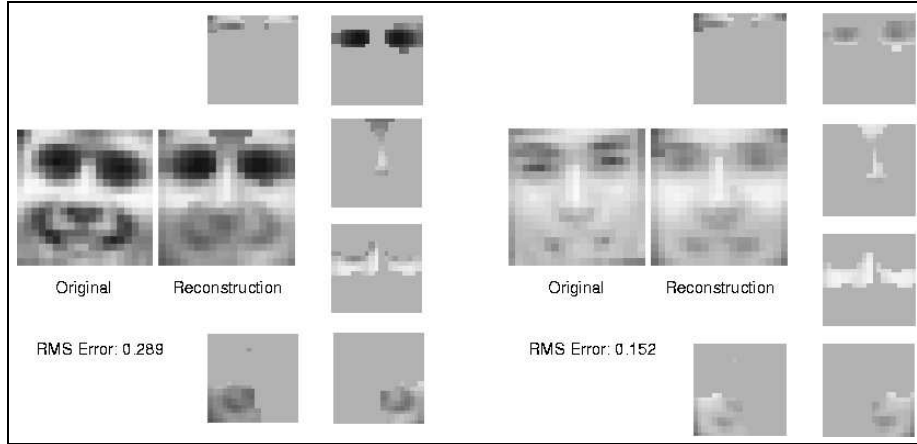

Figure 4: The reconstruction of two test images from the Faces dataset. Beside each reconstruction are the parts—the most active state in each of six VQs—used to generate it. Each part $j \in k$ is represented by its gated prediction ($g_{dk} * m_{kj}$) for each image pixel $i$.

was 0.12 for PCA, 0.20 for NMF, 0.23 for MCVQ (both 3 and 6 VQs), and 0.28 for VQ.

## 5.2 Collaborative Filtering

The application of MCVQ to image data assumes that the images are normalized, i.e., that the head is in a similar pose in each image. Normalization can be difficult to achieve in some image contexts; however, in many other types of applications, the input representation is more stable. For example, many information retrieval applications employ bag-of-words representations, in which a given word always occupies the same input element.

We test MCVQ on a collaborative filtering task, utilizing the EachMovie dataset, where the input vectors are ratings by users of movies, and a given element always corresponds to the same movie. The original dataset contains ratings, on a scale from 1 to 6, of a set of 1649 movies, by 74,424 users. In order to reduce the sparseness of the dataset, since many users rated only a few movies, we only included users who rated at least 75 movies and movies rated by at least 126 users, leaving a total of 1003 movies and 5831 users. The remaining dataset was still very sparse, as the maximum user rated 928 movies, and the maximum movie was rated by 5401 users. We split the data randomly into 4831 users for a training set, and 1000 users in a test set. We ran MCVQ with 8 VQs and 6 states per VQ on this dataset. An example of the results, after 18 iterations of EM, is shown in Fig. 5.

Note that in the MCVQ graphical model (Fig. 1), all the observation dimensions are leaves, so an input variable whose value is not specified in a particular observation vector will not play a role in inference or learning. This makes inference and learning with sparse data rapid and efficient.

We compare the performance of MCVQ on this dataset to the aspect model. We implemented a version of the aspect model, with 50 aspects and truncated Gaussians for ratings, and used "tempered EM" (with smoothing) to fit the parameters[10]. For both models, we train the model on the 4831 users in the training set, and then, for each test user, we let the model observe some fixed number of ratings and hold out the rest. We evaluate the models by measuring the absolute difference between their predictions for a held-out rating and the user's true rating, averaged over all held-out ratings for all test users (Fig. 6).

| The Fugitive 5.8 (6) | Pulp Fiction 5.5 (4) | Cinema Paradiso 5.6 (6) | Shawshank Redemption 5.5 (5) |
|---|---|---|---|
| **Terminator 2** 5.7 (5) | **Godfather: Part II** 5.3 (5) | **Touch of Evil** 5.4 (-) | **Taxi Driver** 5.3 (6) |
| **Robocop** 5.4 (5) | **Silence of the Lambs** 5.2 (4) | **Rear Window** 5.2 (6) | **Dead Man Walking** 5.1 (-) |
| Kazaam 1.9 (-) | Brady Bunch Movie 1.4 (1) | Jean de Florette 2.1 (3) | Billy Madison 3.2 (-) |
| Rent-a-Kid 1.9 (-) | Ready to Wear 1.3 (-) | Lawrence of Arabia 2.0 (3) | Clerks 3.0 (4) |
| Amazing Panda Adventure 1.7 (-) | A Goofy Movie 0.8 (1) | Sense Sensibility 1.6 (-) | Forrest Gump 2.7 (2) |
| **Best of Wallace & Gromit** 5.6 (-) | **Tank Girl** 5.5 (6) | **Mediterraneo** 5.3 (6) | **Sling Blade** 5.4 (5) |
| **The Wrong Trousers** 5.4 (6) | **Showgirls** 5.3 (4) | **Three Colors: Blue** 4.9 (5) | **One Flew ... Cuckoo's Nest** 5.3 (6) |
| **A Close Shave** 5.3 (5) | **Heidi Fleiss...** 5.2 (5) | **Jean de Florette** 4.9 (6) | **Dr. Strangelove** 5.2 (5) |
| Robocop 2.6 (2) | Talking About Sex 2.4 (5) | Jaws 3-D 2.2 (-) | The Beverly Hillbillies 2.0 (-) |
| Dangerous Ground 2.5 (2) | Barbarella 2.0 (4) | Richie Rich 1.9 (-) | Canadian Bacon 1.9 (4) |
| Street Fighter 2.0 (-) | The Big Green 1.8 (2) | Getting Even With Dad 1.5 (-) | Mrs. Doubtfire 1.7 (-) |

Figure 5: The MCVQ representation of two test users in the EachMovie dataset. The 3 most conspicuously high-rated (bold) and low-rated movies by the most active states of 4 of the 8 VQs are shown, where conspicuousness is the deviation from the mean rating for a given movie. Each state's predictions, $\mu_{dkj}$, can be compared to the test user's true ratings (in parentheses); the model's prediction is a convex combination of state predictions. Note the intuitive decomposition of movies into separate VQs, and that different states within a VQ may predict very different rating patterns for the same movies.

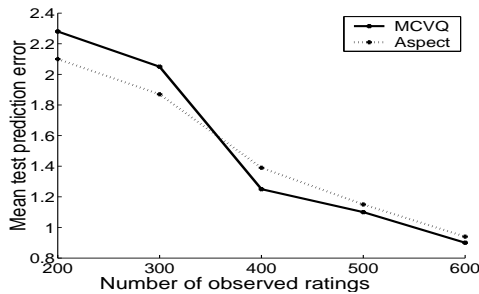

Figure 6: The average absolute deviation of predicted and true values of held-out ratings is compared for MCVQ and the aspect model. Note that the number of users per $x$-bin decreases with increasing $x$, as a user must rate at least $x+1$ movies to be included.

## 5.3 Text Classification

MCVQ can also be used for information retrieval from text documents, by employing the bag-of-words representation. We present preliminary results on the NIPS corpus (available at www.cs.toronto.edu/~roweis/data.html), which consists of the full text of the NIPS conference proceedings, volumes 1 to 12. The data was pre-processed to remove common words (e.g. the), and those appearing in fewer than five documents, resulting in a vocabulary of 14,265 words. For each of the 1740 papers in the corpus, we generated a vector containing the number of occurrences of each word in the vocabulary. These vectors were normalized so that each contained the same number of words. A model of 8 VQs, 8 states each, was trained on the data, converging after 15 iterations of EM. A sample of the results is shown in Fig. 7.

When trained on text data, the values of $\{g_{dk}\}$ provide a segmentation of the vocabulary into subsets of words with correlated frequencies. Within a particular subset, the words can be positively correlated, indicating that they tend to appear in the same documents, or negatively correlated, indicating that they seldom appear together.

## 6 Conclusion

We have presented a novel method for learning factored representations of data which can be efficiently learned, and employed across a wide variety of problem domains. MCVQ combines the cooperative nature of some methods, such as CVQ, NMF, and LSA, that

| Predictive Sequence Learning in Recurrent Neocortical Circuits R. P. N. Rao & T. J. Sejnowski | | | | The Relevance Vector Machine Michael E. Tipping | | | |
|---|---|---|---|---|---|---|---|
| **afferent** | **ekf** | **latent** | **ltp** | **svms** | **hme** | **similarity** | **extraction** |
| **lgn** | **niranjan** | **som** | **gerstner** | **svm** | **svr** | **classify** | **net** |
| **interneurons** | **freitas** | **detection** | **zador** | **margin** | **svs** | **classes** | **weights** |
| **excitatory** | **kalman** | **search** | **soma** | **kernel** | **hyperparameters** | **classification** | **functions** |
| **membrane** | **wp** | **data** | **depression** | **risk** | **kopf** | **class** | **units** |
| query | critic | mdp | spline | jutten | chip | barn | mdp |
| documents | stack | pomdps | tresp | pes | ocular | correlogram | pomdps |
| chess | suffix | prioritized | saddle | cpg | retinal | interaural | littman |
| portfolio | nuclei | singh | hyperplanes | axon | surround | epsp | prioritized |
| players | knudsen | elevator | tensor | behavioural | cmos | bregman | pomdp |

Figure 7: The representation of two documents by an MCVQ model with 8 VQs and 8 states per VQ. For each document we show the states selected for it from 4 VQs. The bold (plain) words for each state are those most conspicuous by their above (below) average predicted frequency.

use multiple causes to generate input, with competitive aspects of clustering methods. In addition, it gains combinatorial power by splitting the input into subsets, and can readily handle sparse, high-dimensional data. One direction of further research involves extending the applications described above, including applying MCVQ to other dimensions of the NIPS corpus such as authors to find groupings of authors based on word-use frequency. An important theoretical direction is to incorporate Bayesian learning for selecting the number and size of each VQ.

## Footnotes

[1] We define description length to be the number of bits required to represent the model, plus the

## References

[1] R.S. Zemel. *A Minimum Description Length Framework for Unsupervised Learning*. PhD thesis, Dept. of Computer Science, University of Toronto, Toronto, Canada, 1993.

[2] G. Hinton and R.S. Zemel. Autoencoders, minimum description length, and Helmholtz free energy. In G. Tesauro J. D. Cowan and J. Alspector, editors, *Advances in Neural Information Processing Systems 6*. Morgan Kaufmann Publishers, San Mateo, CA, 1994.

[3] Z. Ghahramani. Factorial learning and the EM algorithm. In G. Tesauro, D.S. Touretzky, and T.K. Leen, editors, *Advances in Neural Information Processing Systems 7*. MIT Press, Cambridge, MA, 1995.

[4] D.D. Lee and H.S. Seung. Learning the parts of objects by non-negative matrix factorization. *Nature*, 401:788–791, October 1999.

[5] C. Williams and N. Adams. DTs: Dynamic trees. In M.J. Kearns, S.A. Solla, and D.A. Cohn, editors, *Advances in Neural Information Processing Systems 11*. MIT Press, Cambridge, MA, 1999.

[6] G.E. Hinton, Z. Ghahramani, and Y.W. Teh. Learning to parse images. In S.A. Solla, T.K. Leen, and K.R. Muller, editors, *Advances in Neural Information Processing Systems 12*. MIT Press, Cambridge, MA, 2000.

[7] N. Jojic and B.J. Frey. Learning flexible sprites in video layers. In *CVPR*, 2001.

[8] T. Hofmann. Probabilistic latent semantic analysis. In *Proc. of Uncertainty in Artificial Intelligence, UAI'99*, Stockholm, 1999.

[9] D.M. Blei, A.Y. Ng, and M.I. Jordan. Latent Dirichlet allocation. In T.K. Leen, T. Dietterich, and V. Tresp, editors, *Advances in Neural Information Processing Systems 13*. MIT Press, Cambridge, MA, 2001.

[10] T. Hofmann. Learning what people (don't) want. In *European Conference on Machine Learning*, 2001.
